# Sparsity of data representation of optimal kernel machine and leave-one-out estimator

**A. Kowalczyk**
Chief Technology Office, Telstra
770 Blackburn Road, Clayton, Vic. 3168, Australia
(adam.kowalczyk@team.telstra.com)

## Abstract

Vapnik's result that the expectation of the generalisation error of the optimal hyperplane is bounded by the expectation of the ratio of the number of support vectors to the number of training examples is extended to a broad class of kernel machines. The class includes Support Vector Machines for soft margin classification and regression, and Regularization Networks with a variety of kernels and cost functions. We show that key inequalities in Vapnik's result become equalities once "the classification error" is replaced by "the margin error", with the latter defined as an instance with positive cost. In particular we show that expectations of the true margin error and the empirical margin error are equal, and that the sparse solutions for kernel machines are possible only if the cost function is "partially" insensitive.

## 1 Introduction

Minimization of regularized risk is a backbone of several recent advances in machine learning, including Support Vector Machines (*SVM*) [13], Regularization Networks (*RN*) [5] or Gaussian Processes [15]. Such a machine is typically implemented as a weighted sum of a kernel function evaluated for pairs composed of a data vector in question and a number of selected training vectors, so called support vectors. For practical machines it is desired to have as few support vectors as possible. It has been observed empirically that SVM solutions have often very few support vectors, or that they are *sparse*, while RN machines are not. The paper shows that this behaviour is determined by the properties of the cost function used (its partial insensitivity, to be precise).

Another motivation for interest in sparsity of solutions comes from celebrated result of Vapnik [13] which links the number of support vectors to the generalization error of SVM via a bound on leave-one-out estimator [9]. This result has been originally shown for a special case of classification with hard margin cost function (optimal hyperplane). The papers by Opper and Winther [10], Jaakkola and Haussler [6], and Joachims [7] extend Vapnik's result in the direction of bounds for classification error of SVM's. The first of those papers deals with the hard margin case, while the other two derive tighter bounds on classification error of the soft margin SVMs with $\epsilon$-insensitive linear cost.

In this paper we extend Vapnik's result in another direction. Firstly, we show that it holds for to a wide range of kernel machines optimized for a variety of cost functions, for both

classification and regression tasks. Secondly, we find that Vapnik's key inequalities become equalities once "the misclassification error" is replaced by "the margin error" (defined as the rate of data instances incurring positive costs). In particular, we find that for margin errors the following three expectations : (i) of the empirical risk, (ii) of the the true risk and (iii) of the leave-one-out risk estimator are equal to each other. Moreover, we show that they are equal to the expectation of the ratio of support vectors to the number of training examples.

The main results are given in Section 2. Brief discussion of results is given in Section 3.

## 2   Main results

Given an $l$-sample $\{(x_1, y_1), ...., (x_l, y_l)\}$ of patterns $x_i \in X \subset \mathbb{R}^n$ and target values $y_i \in Y \subset \mathbb{R}$. The learning algorithms used by SVMs [13], RNs [5] or Gaussian Processes [15] minimise the regularized risk functional of the form:

$$\min_{(f,b) \in \mathcal{H} \times \mathbb{R}} R_{reg}[f, b] = \sum_{i=1}^{l} c(x_i, y_i, \xi_i[f, b]) + \frac{\lambda}{2} \|f\|_{\mathcal{H}}^2. \tag{1}$$

Here $\mathcal{H}$ denotes a reproducing kernel Hilbert space (RKHS) [1], $\|.\|_{\mathcal{H}}$ is the corresponding norm, $\lambda > 0$ is a regularization constant, $c : X \times Y \times \mathbb{R} \to \mathbb{R}^+$ is a *non-negative* cost function penalising for the deviation $\xi_i[f, b] = y_i - \hat{y}_i$ of the estimator $\hat{y}_i := f(x_i) + \beta b$ from target $y_i$ at location $x_i$, $b \in \mathbb{R}$ is a constant (bias) and $\beta \in \{0, 1\}$ is another constant ($\beta = 0$ is used to switch the bias off).

The important Representer Theorem [8, 4] states that the minimizer (1) has the expansion:

$$f(x) = \sum_{i=1}^{l} \alpha_i k(x_i, x), \tag{2}$$

where $k : X \times X \to \mathbb{R}$ is the kernel corresponding to the RKHS $\mathcal{H}$. In the following section we shall show that under general assumptions this expansion is unique.

If $\alpha_i \neq 0$, then $x_i$ is called *a support vector* of $f(.)$.

### 2.1   Unique Representer Theorem

We recall, that a function is called *a real analytic function* on a domain $\subset \mathbb{R}^q$ if for every point of this domain the Taylor series for the function converges to this function in some neighborhood of that point.[1]

A proof of the following crucial Lemma is omitted due to lack of space.

**Lemma 2.1.** *If $\varphi : X \to \mathbb{R}$ is an analytic function on an open connected subset $X \subset \mathbb{R}^n$, then the subset $\varphi^{-1}(0) \subset X$ is either equal to $X$ or has Lebesgue measure 0.*

Analyticity is essential for the above result and the result does not hold even for functions infinitely differentiable, in general. Indeed, for every closed subset $V \subset \mathbb{R}^n$ there exists an infinitely differentiable function ($C^\infty$) on $\mathbb{R}^n$ such that $\phi^{-1}(0) = V$ and there exist closed subsets with positive Lebesgue measure and empty interior. Hence the Lemma, and consequently the subsequent results, do not hold for the broader class of all $C^\infty$ functions.

**Standing assumptions.** The following is assumed.

1. The set $X \subset \mathbb{R}^n$ is open and connected and either $Y = \{\pm 1\}$ (the case of *classification*) or $Y \subset \mathbb{R}$ is an open segment (the case of *regression*).

2. The kernel $k : X \times X \to \mathbb{R}$ is a real analytic function on its domain.

3. The cost function $\xi \mapsto c(x, y, \xi)$ is convex, differentiable on $\mathbb{R}$ and $c(x, y, 0) = 0$ for every $(x, y) \in X \times Y$. It can be shown that

$$c(x, y, \xi) > 0 \quad \Leftrightarrow \quad \frac{\partial c}{\partial \xi}(x, y, \xi) \neq 0. \tag{3}$$

4. $l$ is a fixed integer, $1 < l \leq \dim(\mathcal{H})$, and the training sample $(x_1, y_1), ..., (x_l, y_l)$ is iid drawn from a continuous probability density $p(x, y)$ on $X \times Y$.

5. The phrase "with probability 1" will mean with probability 1 with respect to the selection of the training sample.

Note that standard polynomial kernel $k(x, x') = (1 + x \cdot x')^d$, $x, x' \in \mathbb{R}^n$, satisfies the above assumptions with $\dim(\mathcal{H}) = \binom{n+d}{d}$. Similarly, the Gaussian kernel $k(x, x') = \exp(-||x - x'||^2/\sigma)$ satisfies them with $\dim(\mathcal{H}) = \infty$.

Typical cost functions such as the super-linear loss functions $c_p(x, y, \xi) = (y\xi)_+^p := (\max(0, y\xi))^p$ used for SVM classification, or $c_{p\epsilon}(x, y, \xi) = (|\xi| - \epsilon)_+^p$ used for SVM regression, or the super-linear loss $c_p(x, y, \xi) = |\xi|^p$ for $p > 1$ for RN regression, satisfy the above assumptions[2]. Similarly, variations of Huber robust loss [11, 14] satisfy those assumptions.

The following result strengthens the Representer Theorem [8, 4]

**Theorem 2.2.** *If $l \leq \dim \mathcal{H}$, then both, the minimizer of the regularized risk (1) and its expansion (2) are unique with probability 1.*

**Proof outline.** Convexity of the functional $(f, b) \mapsto R_{reg}[f, b]$ and its strict convexity with respect to $f \in \mathcal{H}$ implies the uniqueness of $f \in \mathcal{H}$ minimizing the regularized risk (1); cf.[3]. From the assumption that $l \leq \dim \mathcal{H}$ we derive the existence of $\tilde{x}_1, ..., \tilde{x}_l \in X$ such that the functions $f(\tilde{x}_i, .)$, $i = 1, ..., l$, are linearly independent. Equivalently, the following Gram determinant is $\neq 0$:

$$\phi(\tilde{x}_1, ..., \tilde{x}_l) := \det[\langle k(\tilde{x}_i, .), k(\tilde{x}_j, .) \rangle_{\mathcal{H}}]_{1 \leq i,j \leq l} = \det[k(\tilde{x}_i, \tilde{x}_j)]_{1 \leq i,j \leq l} \neq 0.$$

Now Lemma 2.1 implies that $\phi(x_1, ..., x_l) \neq 0$ with probability 1, since $\phi : X^l \to \mathbb{R}$ is an analytic function. Hence functions $k(x_i, .)$ are linearly independent and the expansion (2) is unique with probability 1. Q.E.D.

## 2.2 Leave-one-out estimator

In this section the minimizer (1) for the whole data sequence of $l$-training instances and some other objects related to it will be additionally marked with superscript '$(l)$'. The superscript '$(l\backslash i)$' will be used analogously to mark objects corresponding to the minimizer of (1) for the reduced training sequence, with $i$th instance removed.

**Lemma 2.3.** *With probability 1, for every $i \in \{1, ..., l\}$:*

$$\alpha^{(l)}_i \neq 0 \quad \Leftrightarrow \quad c(x_i, y_i, \xi_i[f^{(l)}, b^{(l)}]) > 0, \tag{4}$$

$$\alpha^{(l)}_i \neq 0 \quad \Leftrightarrow \quad c(x_i, y_i, \xi_i[f^{(l\backslash i)}, b^{(l\backslash i)}]) > 0. \tag{5}$$

**Proof outline.** With probability 1, functions $k(x_j, .)$, $j = 1, ..., l$, are linearly independent (cf. the proof of Theorem 2.2) and there exists a feature map $\Phi : X \to \mathbb{R}^l$ such that vectors $z_j := \Phi(x_j)$, $i = 1, ..., l$ are linearly independent, $k(x_j, x) = z_j \cdot \Phi(x)$ and $f^{(l)}(x) = z^{(l)} \cdot \Phi(x) + \beta b^{(l)}$ for every $x \in X$, where $z^{(l)} := \sum_{j=1}^{l} \alpha^{(l)}{}_j z_j$. The pair $(z^{(l)}, b^{(l)})$ minimizes the function

$$\tilde{R}_{reg}^{(l)}(z, b) := \sum_{j=1}^{l} c(x_j, y_j, \tilde{\xi}_j(z, b)) + \frac{\lambda}{2}\|z\|^2 \tag{6}$$

where $\tilde{\xi}_j(z, b) := y_j - z \cdot z_j - \beta b$. This function is differentiable due to the standing assumptions on the cost $c$. Hence, necessarily $\text{grad}R_{reg} = 0$, at the minimum $(z^{(l)}, b^{(l)})$, which due to the linear independence of vectors $z_j$, gives

$$\alpha^{(l)}{}_j = -\frac{1}{\lambda} \frac{\partial c}{\partial \xi}(x_j, y_j, \tilde{\xi}_j(z^{(l)}, b^{(l)})) \tag{7}$$

for every $j = 1, ..., l$. This equality combined with equivalence (3) proves (4).

Now we proceed to the proof of (5). Note that the pair $(z^{(l \backslash i)}, b^{(l \backslash i)})$, where $z^{(l \backslash i)} := \sum_{j \neq i}^{l} \alpha^{(l \backslash i)}{}_j z_j$, corresponds in the feature space to the minimizer $(f^{(l \backslash i)}, b^{(l \backslash i)})$ of the reduced regularized risk:

$$\tilde{R}_{reg}^{(l \backslash i)}(z, b) := \sum_{j=1;\ j \neq i}^{l} c(x_j, y_j, \tilde{\xi}_j(z, b)) + \frac{\lambda}{2}\|z\|^2.$$

*Sufficiency in* (5). From (4) and characterization (7) of the critical point it follows immediately that if $\alpha^{(l)}{}_i = 0$, then the minimizers for the full and reduced data sets are identical.

*Necessity in* (5). A supposition of $\alpha^{(l)}{}_i \neq 0$ and $c(x_i, y_i, \xi_i[f^{(l \backslash i)}, b^{(l \backslash i)}]) = 0$ leads to a contradiction. Indeed, from (4), $c(x_i, y_i, \tilde{\xi}_i(z^{(l)}, b^{(l)})) > 0$, hence:

$$
\begin{aligned}
\tilde{R}_{reg}^{(l)}(z^{(l \backslash i)}, b^{(l \backslash i)}) &= \tilde{R}_{reg}^{(l \backslash i)}(z^{(l \backslash i)}, b^{(l \backslash i)}) \\
&\leq \tilde{R}_{reg}^{(l \backslash i)}(z^{(l)}, b^{(l)}) = \tilde{R}_{reg}^{(l)}(z^{(l)}, b^{(l)}) - c(x_i, y_i, \tilde{\xi}_i(z^{(l)}, b^{(l)})) \\
&< \tilde{R}_{reg}^{(l)}(z^{(l)}, b^{(l)}) = \min_{(z,b) \in \mathbb{R}^l \times \mathbb{R}} R_{reg}^{(l)}(z, b).
\end{aligned}
$$

This contradiction completes the proof. Q.E.D.

We say that $x_i$ is *a sensitive support vector* if $\alpha^{(l)}{}_i \neq 0$ and $f^{(l)} \neq f^{(l \backslash i)}$, i.e., if its removal from the training set changes the solution.

**Corollary 2.4.** *Every support vector is sensitive with probability 1.*

**Proof.** If $\alpha_i \neq 0$, then the vector $z^{(l)} \notin \text{Lin}_{\mathbb{R}}(z_1, ...., z_{i-1}, z_{i+1}, ..., z_l)$ since $z^{(l)}$ has a non-trivial component $\alpha_i z_i$ in the direction of $i$th feature vector $z_i$, while $z^{(l \backslash i)} \in \text{Lin}_{\mathbb{R}}(z_1, ...., z_{i-1}, z_{i+1}, ..., z_l)$. Thus $z^{(l)}$ and $z^{(l \backslash i)}$ have different directions in $\text{Lin}_{\mathbb{R}}(z_1, ..., z_l) \subset Z$ and there exists $j' \in \{1, ..., l\}$ such that $f^{(l)}(x_{j'}) \neq f^{(l \backslash i)}(x_{j'})$. Q.E.D.

We define the *empirical risk* and *the expected (true) risk* of margin error

$$
\begin{aligned}
R_{emp}[f, b] &:= \frac{\sum_{i=1}^{l} I_{\{c(x_i, y_i, \xi_i[f, b]) > 0\}}}{l} = \frac{\#\{i\ ;\ c(x_i, y_i, \xi_i[f, b]) > 0\}}{l}, \\
R_{exp}[f, b] &:= \text{Prob}[c(x, y, y - f(x) - \beta b) > 0],
\end{aligned}
$$

where $(f, b) \in \mathcal{H} \times \mathbb{R}$, $I_{\{.\}}$ denotes the indicator function and $\#$ denotes the cardinality (number of elements) of a set.

From the above Lemma we obtain immediately the following result:

**Corollary 2.5.** *With probability 1:*

$$\frac{\#\{i \; ; \; c(x_i, y_i, f^{(l\backslash i)}(x_i) + \beta b^{(l\backslash i)}) > 0\}}{l} = \frac{\#\{i \; ; \; \alpha^{(l)}_i \neq 0\}}{l} = R_{emp}[f^{(l)}, b^{(l)}].$$

There exist counter-examples showing the phrase "with probability 1" above cannot be omitted. The sum on L.H.S. above is the *leave-one-out estimator* of the risk of margin error [14] for the minimizer of regularized risk (1). The above corollary shows that this estimator is uniquely determined by the number of support vectors as well as the number of training margin errors.

Now from the Lunts-Brailovsky Theorem [14, Theorem 10.8] applied to the risk $Q(x, y; f, b) := I_{\{c(x,y,y-f(x)-\beta b > 0\}}$ the following result is obtained.

**Theorem 2.6.**

$$\mathbf{E}[R_{exp}(f^{(l-1)}, b^{(l-1)})] = \mathbf{E}[R_{emp}(f^{(l)}, b^{(l)})] = \frac{\mathbf{E}[\#\{i \; ; \; \alpha^{(l)}_i \neq 0\}]}{l}, \tag{8}$$

*where the first expectation is in the selection of training $(l - 1)$-sample and the remaining two are with respect to the selection of training $l$-sample.*

A cost function is called *partially insensitive* if there exists $(x, y) \in X \times Y$ and $\xi_1 \neq \xi_2$ such that $c(x, y, \xi_1) = c(x, y, \xi_2) = 0$. Otherwise, the cost $c$ is called *sensitive*. Typical SVM cost functions are partially insensitive while typical RN cost functions are sensitive. The following result can be derived from Theorem 2.6 and Lemma 2.3.

**Corollary 2.7.** *If the number of support vectors is $< l$ with a probability $> 0$, then the cost function has to be partially insensitive.*

Typical cost functions penalize for an allocation of a wrong sign, i.e.

$$\forall_{(x,y,\hat{y}) \in X \times Y \times \mathbb{R}} \quad y\hat{y} < 0 \quad \Rightarrow \quad c(x, y, y - \hat{y}) > 0. \tag{9}$$

Let us define *the risk of misclassification* of the kernel machine $\hat{y}(x) = f(x) + \beta b$ for $(f, b) \in \mathcal{H} \times \mathbb{R}$ as $R_{clas}[f, b] := \text{Prob}[y\hat{y}(x) < 0]$. Assuming (9), we have $R_{clas}[f, b] \leq R_{exp}[f, b]$. Combining this observation with (8) we obtain an extension of Vapnik's result [14, Theorem 10.5]:

**Corollary 2.8.** *If condition (9) holds then*

$$\mathbf{E}[R_{clas}(f^{(l-1)}, b^{(l-1)})] \leq \frac{\mathbf{E}[\#\{i \; ; \; \alpha^{(l)}_i \neq 0\}]}{l} = \mathbf{E}[R_{emp}(f^{(l)}, b^{(l)})]. \tag{10}$$

Note that the original Vapnik's result consists in an inequality analogous to the inequality in the above condition for the specific case of classification by optimal hyperplanes (hard margin support vector machines).

## 3  Brief Discussion of Results

**Essentiality of assumptions.** For every formal result in this paper and any of the standing assumption there exists an example of a minimizer of (1) which violates the conclusions of the result. In this sense all those assumptions are essential.

**Linear combinations of admissible cost functions.** Any weighted sum of cost functions satisfying our Standing Assumption 3 will satisfy this assumption as well, hence our

formalism will apply to it. An illustrative example is the following cost function for classification $c(x, y, \xi) = \sum_j^q C_j (\max(0, y(\xi - \epsilon_j)))^{p_j}$, where $C_j > 0$, $\epsilon_j \geq 0$ and $p_j > 1$ are constants and $y \in Y = \{\pm 1\}$.

**Non-differentiable cost functions.** Our formal results can be extended with minor modifications to the case of typical, non-differentiable *linear cost function* such as $c = (y\xi)_+ = \max(0, y\xi)$ for SVM classification, $c = (|\xi| - \epsilon)_+$ for SVM regression and to the classification with hard margins SVMs (optimal hyperplanes). Details are beyond the scope of this paper. Note that the above linear cost functions can be uniformly approximated by differentiable cost functions, e.g. by Huber cost function [11, 14], to which our formalism applies. This implies that our formalism "applies approximately" to the linear loss case and some partial extension of it can be obtained directly using some limit arguments. However, using direct algebraic approach based on an evaluation of Kuhn-Tucker conditions one can come to stronger conclusions. Details will be presented elsewhere.

**Theory of generalization.** Equality of expectations of empirical and expected risk provided by Theorem 2.6 implies that minimizers of regularized risk (1) are on average consistent. We should emphasize that this result holds for small training samples, of the size $l$ smaller than VC dimension of the function class, which is $\dim(\mathcal{H}) + 1$ in our case. This should be contrasted with uniform convergence bounds [2, 13, 14] which are vacuous unless $l >> $ VC dimension.

**Significance of approximate solutions for RNs.** Corollary 2.7 shows that sparsity of solutions is practically not achievable for optimal RN solutions since they use sensitive cost functions. This emphasizes the significance of research into approximately optimal solution algorithms in such a case, cf. [12].

**Application to selection of the regularization constant.** The bound provided by Corollary 2.8 and the equivalence given by Theorem 2.6 can be used as a justification of a heuristic that the optimal value of regularization constant $\lambda$ is the one which minimizes the number of margin errors (cf. [14]). This is especially appealing in the case of regression with $\epsilon$-insensitive cost, where the margin error has a straightforward interpretation of sample being outside of the $\epsilon$-tube.

**Application to modelling of additive noise.** Let us suppose that data is iid drawn form the distribution of the form $y = f(x) + \epsilon_{noise}$, where $\epsilon_{noise}$ is a random noise independent of $x$, with 0 mean. Theorem 2.6 implies the following heuristic for approximation of the noise distribution in the regression model $y = f(x) + \epsilon_{noise}$:

$$\text{Prob}[\epsilon_{noise} > \epsilon] \approx \frac{\#\{i \; ; \; \alpha^{(l)} \neq 0\}}{l}.$$

Here $(f^{(l)}, b^{(l)})$ is a minimizer of the regularized risk (1) with an $\epsilon$-insensitive cost function, i.e. such that $c(x, y, \xi) > 0$ iff $|\xi| > \epsilon$.

**Acknowledgement.** The permission of the Chief Technology Officer, Telstra, to publish this paper, is gratefully acknowledged.

## Footnotes

[1]Examples of analytic functions are polynomials. The ordinary functions such as $\sin(x)$, $\cos(x)$ and $\exp(x)$ are examples of non-polynomial analytic functions. The function $\psi(x) := \exp(-1/x^2)$ for $x > 0$ and 0, otherwise, is an example of infinitely differentiable function of the real line but not analytic (locally it is not equal to its Taylor series expansion at zero).

[2]Note that in general, if a function $\phi : \mathbb{R} \to \mathbb{R}$ is convex, differentiable and such that $d\phi/d\xi(0) = 0$, then the cost function $c(x, y, \xi) := \phi((\xi)_+)$ is convex and differentiable.

# References

[1] N. Aronszajn. Theory of reproducing kernels. *Transactions of the American Mathematical Society*, 68:337 – 404, 1950.

[2] P. Bartlett and J. Shave-Taylor. Generalization performance of support vector machines and other pattern classifiers. In B. Schölkopf, *et. al.*, eds., *Advances in Kernel Methods*, pages 43–54, MIT Press, 1998.

[3] C. Burges and D. J. Crisp. Uniqueness of the SVM solution. In S. Sola *et. al.*, ed., *Adv. in Neural Info. Proc. Sys. 12*, pages 144–152, MIT Press, 2000.

[4] D. Cox and F. O'Sullivan. Asymptotic analysis of penalized likelihood and related estimators. *Ann. Statist.*, 18:1676–1695, 1990.

[5] F. Girosi, M. Jones, and T. Poggio. Regularization theory and neural networks architectures. *Neural Computation*, 7(2):219–269, 1995.

[6] T. Jaakkola and D. Haussler. Probabilistic kernel regression models. In *Proc. Seventh Work. on AI and Stat.*, San Francisco, 1999. Morgan Kaufman.

[7] T. Joachims. Estimating the Generalization Performance of an SVM Efficiently. In *Proc. of the International Conference on Machine Learning*, 2000. Morgan Kaufman.

[8] G. Kimeldorf and G. Wahba. A correspondence between Bayesian estimation of stochastic processes and smoothing by splines. *Ann. Math. Statist.*, 41:495–502, 1970.

[9] A. Lunts and V. Brailovsky. Evaluation of attributes obtained in statistical decision rules. *Engineering Cybernetics*, 3:98–109, 1967.

[10] M. Opper and O. Winther. Gaussian process classification and SVM: Mean field results and leave-one out estimator. In P. Bartlett, *et. al* eds., *Advances in Large Margin Classifiers*, pages 301–316, MIT Press, 2000.

[11] A. Smola and B. Schölkopf. A tutorial on support vector regression. *Statistics and Computing*, 1998. In press.

[12] A. J. Smola and B. Schölkopf. Sparse greedy matrix approximation for machine learning. Typescript, March 2000.

[13] V. Vapnik. *The Nature of Statistical Learning Theory*. Springer Verlag, New York, 1995.

[14] V. Vapnik. *Statistical Learning Theory*. Wiley, New York, 1998.

[15] C. K. I. Williams. Prediction with Gaussian processes: From linear regression to linear prediction and beyond. In M. I. Jordan, editor, *Learning and Inference in Graphical Models*. Kluwer, 1998.
